# SONG LEARNING IN BIRDS

M. Konishi
Division of Biology
California Institute of Technology

Birds sing to communicate. Male birds use song to advertise their territories and attract females. Each bird species has a unique song or set of songs. Song conveys both species and individual identity. In most species, young birds learn some features of adult song. Song develops gradually from amorphous to fixed patterns of vocalization as if crystals form out of liquid. Learning of a song proceeds in two steps; birds commit the song to memory in the first stage and then they vocally reproduce it in the second stage. The two stages overlap each other in some species, while they are separated by several months in other species. The ability of a bird to commit a song to memory is restricted to a period known as the sensitive phase. Vocal reproduction of the memorized song requires auditory feedback. Birds deafened before the second stage cannot reproduce the memorized song. Birds change vocal output until it matches with the memorized song, which thus serves as a template. Birds use a built-in template when a tutor model is not available. Exposure to a tutor model modifies this innate template.

A series of brain nuclei controls song production and patterning. Recording multi- and single neurons from this nuclei in the singing bird is possible. The learned temporal pattern of song is recognizable in the neural discharge of these nuclei. The need for auditory feedback for song learning suggests the presence of links between the auditory and vocal control systems. One such link is found in the HVc, one of the forebrain song nuclei. This nucleus contains neurons sensitive to sound in addition to those which control song production. In the white-crowned sparrow, the HVc contains neurons selective for the bird's own individual song. The stimulus selectivity of these neurons are thus shaped by the bird's hearing of its own voice during song development.

[1] Konishi, M. (1985) Birdson: from behavior to neuron. *Ann. Rev. Neurosci.* 8:125-170.

[2] Konishi, M. (1985) The role of auditory feedback in the control of vocalization in the white-crowned sparrow. *Z. Tierpsychol.* 22:770-783.

[3] McCasland, J. S. (1987) Neuronal control of bird song production. *J. Neurosci.*, 723-739.

[4] Margoliash, D. (1983) Acoustic parameters underlying the responses of song-specific neurons in the white-crowned sparrow. *J. Neurosci.* 3:10389-1057.

[5] Nottebohm, F. T., Stokes, M., & Leonard, C. M. (1976) Central control of song in the canary Serinus canarius. *J. Comp. Neurol.* 165:457-486.
